# Multi-Digit Recognition Using A Space Displacement Neural Network

Ofer Matan*, Christopher J.C. Burges,
Yann Le Cun and John S. Denker
AT&T Bell Laboratories, Holmdel, N. J. 07733

## Abstract

We present a feed-forward network architecture for recognizing an unconstrained handwritten multi-digit string. This is an extension of previous work on recognizing isolated digits. In this architecture a single digit recognizer is replicated over the input. The output layer of the network is coupled to a Viterbi alignment module that chooses the best interpretation of the input. Training errors are propagated through the Viterbi module.

The novelty in this procedure is that segmentation is done on the feature maps developed in the Space Displacement Neural Network (SDNN) rather than the input (pixel) space.

## 1 Introduction

In previous work (Le Cun et al., 1990) we have demonstrated a feed-forward back-propagation network that recognizes isolated handwritten digits at state-of-the-art performance levels. The natural extension of this work is towards recognition of unconstrained *strings* of handwritten digits. The most straightforward solution is to divide the process into two: segmentation and recognition. The segmenter will divide the original image into pieces (each containing an isolated digit) and pass it to the recognizer for scoring. This approach assumes that segmentation and recognition can be decoupled. Except for very simple cases this is not true.

Speech-recognition research (Rabiner, 1989; Franzini, Lee and Waibel, 1990) has demonstrated the power of using the recognition engine to score each segment in

a candidate segmentation. The segmentation that gives the best combined score is chosen. "Recognition driven" segmentation is usually used in conjunction with dynamic programming, which can find the optimal solution very efficiently.

Though dynamic programming algorithms save us from exploring an exponential number of segment combinations, they are still linear in the number of possible segments – requiring one call to the recognition unit per candidate segment. In order to solve the problem in reasonable time it is necessary to: 1) limit the number of possible segments, or 2) have a rapid recognition unit.

We have built a ZIP code reading system that "prunes" the number of candidate segments (Matan et al., 1991). The candidate segments were generated by analyzing the image's pixel projection onto the horizontal axis. The strength of this system is that the number of calls to the recognizer is small (only slightly over twice the number of real digits). The weakness is that by generating only a small number of candidates one often misses the correct segmentation. In addition, generation of this small set is based on multi-parametric heuristics, making tuning the system difficult.

It would be attractive to discard heuristics and generate many more candidates, but then the time spent in the recognition unit would have to be reduced considerably. Reducing the computation of the recognizer usually gives rise to a reduction in recognition rates. However, it is possible to have our segments and eat them too. We propose an architecture which can explore many more candidates without compromising the richness of the recognition engine.

## 2   The Design

Let us describe a simplified and less efficient solution that will lead us to our final design. Consider a de-skewed image such as the one shown in Figure 1. The system will separate it into candidate segments using vertical cuts. A few examples of these are shown beneath the original image in Figure 1. In the process of finding the best overall segmentation each candidate segment will be passed to the recognizer described in (Le Cun et al., 1990). The scores will be converted to probabilities (Bridle, 1989) that are inserted into nodes of a direct acyclic graph. Each path on this graph represents a candidate segmentation where the length of each path is the product of the node values along it. The Viterbi algorithm is used to determine the longest path (which corresponds to the segmentation with the highest combined score).

It seems somewhat redundant to process the same pixels numerous times (as part of different, overlapping candidate segments). For this reason we propose to pass a whole size-normalized image to the recognition unit and to segment a feature map, after most of the neural network computation has been done. Since the first four layers in our recognizer are convolutional, we can easily extend the single-digit network by applying the convolution kernels to the multi-digit image.

Figure 2 shows the example image (Figure 1) processed by the extended network. We now proceed to segment the top layer. Since the network is convolutional, segmenting this feature-map layer is similar to segmenting the input layer. (Because of overlapping receptive fields and reduced resolution, it is not exactly equivalent.) This gives a speed-up of roughly an order of magnitude.

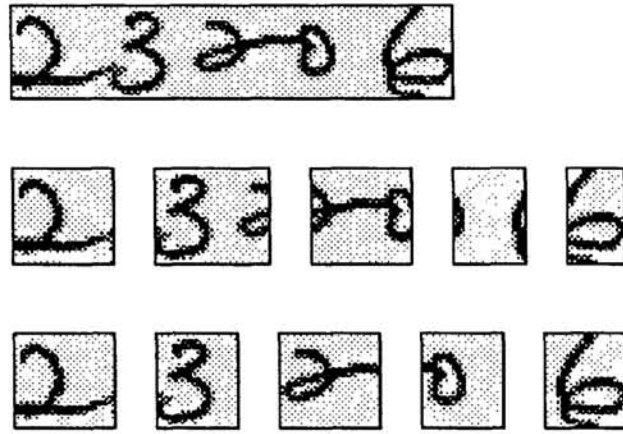

Figure 1: A sample ZIP code image and possible segmentations.

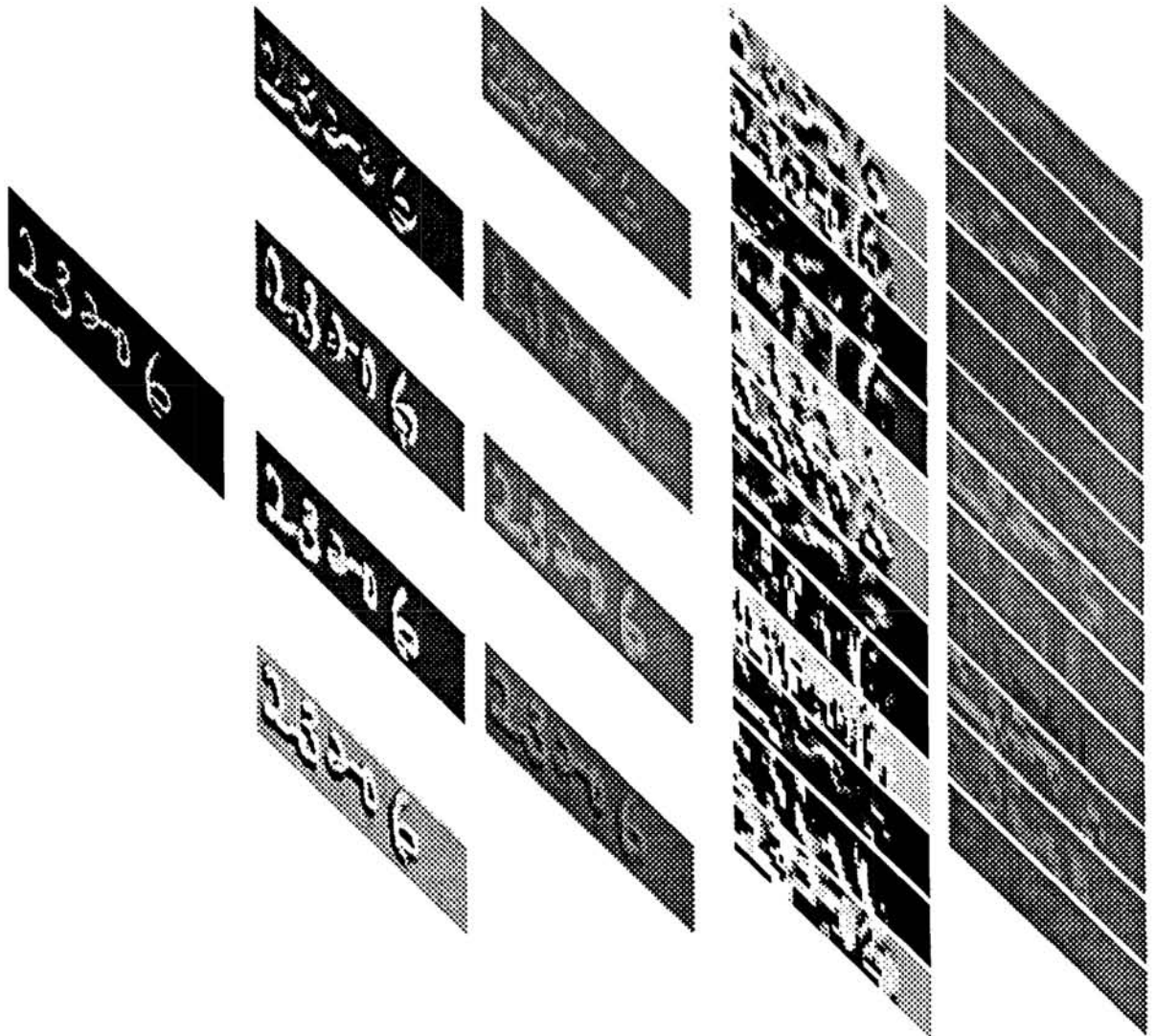

Figure 2: The example ZIP code processed by 4 layers of a convolutional feed-forward network.

In the single digit network, we can view the output layer as a 10-unit column vector that is connected to a zone of width 5 on the last feature layer. If we replicate the single digit network over the input in the horizontal direction, the output layer will be replicated. Each output vector will be connected to a different zone of width 5 on the feature layer. Since the width of a handwritten digit is highly variable, we construct alternate output vectors that are connected to feature segment zones of widths 4,3 and 2. The resulting output maps for the example ZIP code are shown in Figure 3.

The network we have constructed is a shared weight network reminiscent of a TDNN (Lang and Hinton, 1988). We have termed this architecture a Space Displacement Neural Network (SDNN). We rely on the fact that most digit strings lie on more or less one line; therefore, the network is replicated in the horizontal direction. For other applications it is conceivable to replicate in the vertical direction as well.

## 3   The Recognition Procedure

The output maps are processed by a Viterbi algorithm which chooses the set of output vectors corresponding to the segmentation giving the highest combined score. We currently assume that we know the number of digits in the image; however, this procedure can be generalized to an unknown number of digits. In Figure 3 the five output vectors that combined to give the best overall score are marked by thin lines beneath them.

## 4   The Training Procedure

During training we follow the above procedure and repeat it under the constraint that the winning combination corresponds to the ground truth. In Figure 4 the constrained-winning output vectors are marked by small circles. We perform back-propagation through both the ground truth vectors (reinforcement) and highest scoring vectors (negative reinforcement).

We have trained and tested this architecture on size normalized 5-digit ZIP codes taken from U.S Mail. 6000 images were used for training and 3000 where used for testing. The images were cleaned, deskewed and height normalized according to the assumed largest digit height. The data was not "cleaned" after the automatic preprocessing, leaving non centered images and non digits in both the training and test set.

Training was done using stochastic back propagation with some sweeps using Newton's method for adjusting the learning rates. We tried various methods of initializing the gradient on the last layer:

- Reinforce only units picked by the constrained Viterbi. (all other units have a gradient of zero).

- Same as above, but set negative feedback through units chosen by regular Viterbi that are different from those chosen by the constrained version. (Push down the incorrect segmentation if it is different from the correct answer). This speeds up the convergence.

- Reinforce units chosen by the constrained Viterbi. Set negative feed back

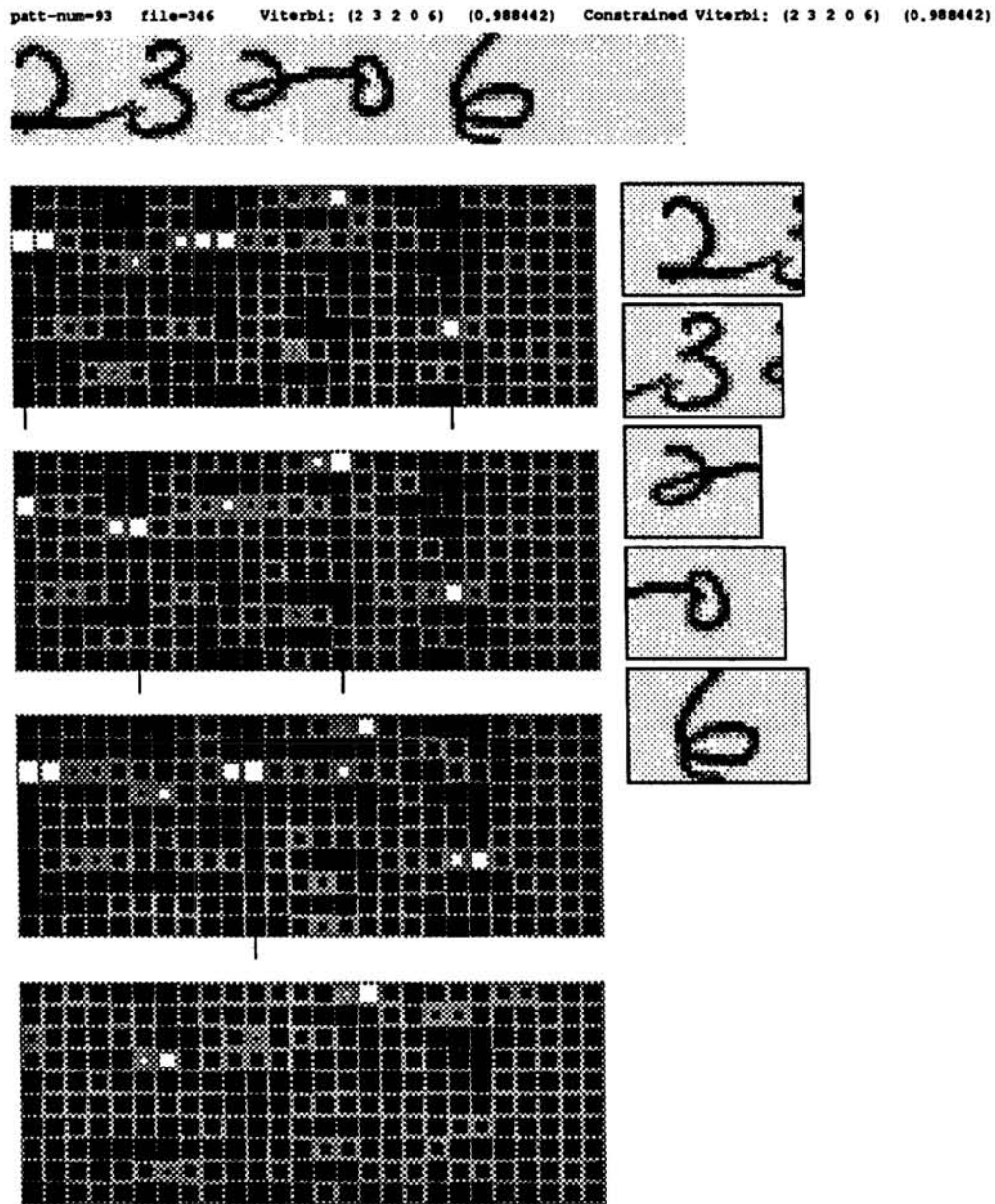

Figure 3: Recognition using the SDNN/Viterbi. The output maps of the SDNN are shown. White indicates a positive activation. The output vectors chosen by the Viterbi alignment are marked by a thin line beneath them. The input regions corresponding to these vectors are shown. One can see that the system centers on the individual digits. Each of the 4 output maps shown is connected to different size zone in the last feature layer (5,4,3 and 2, top to bottom). In order to implement weight sharing between output units connected to different zone sizes, the dangling connections to the output vectors of narrower zones are connected to feature units corresponding to background in the input.

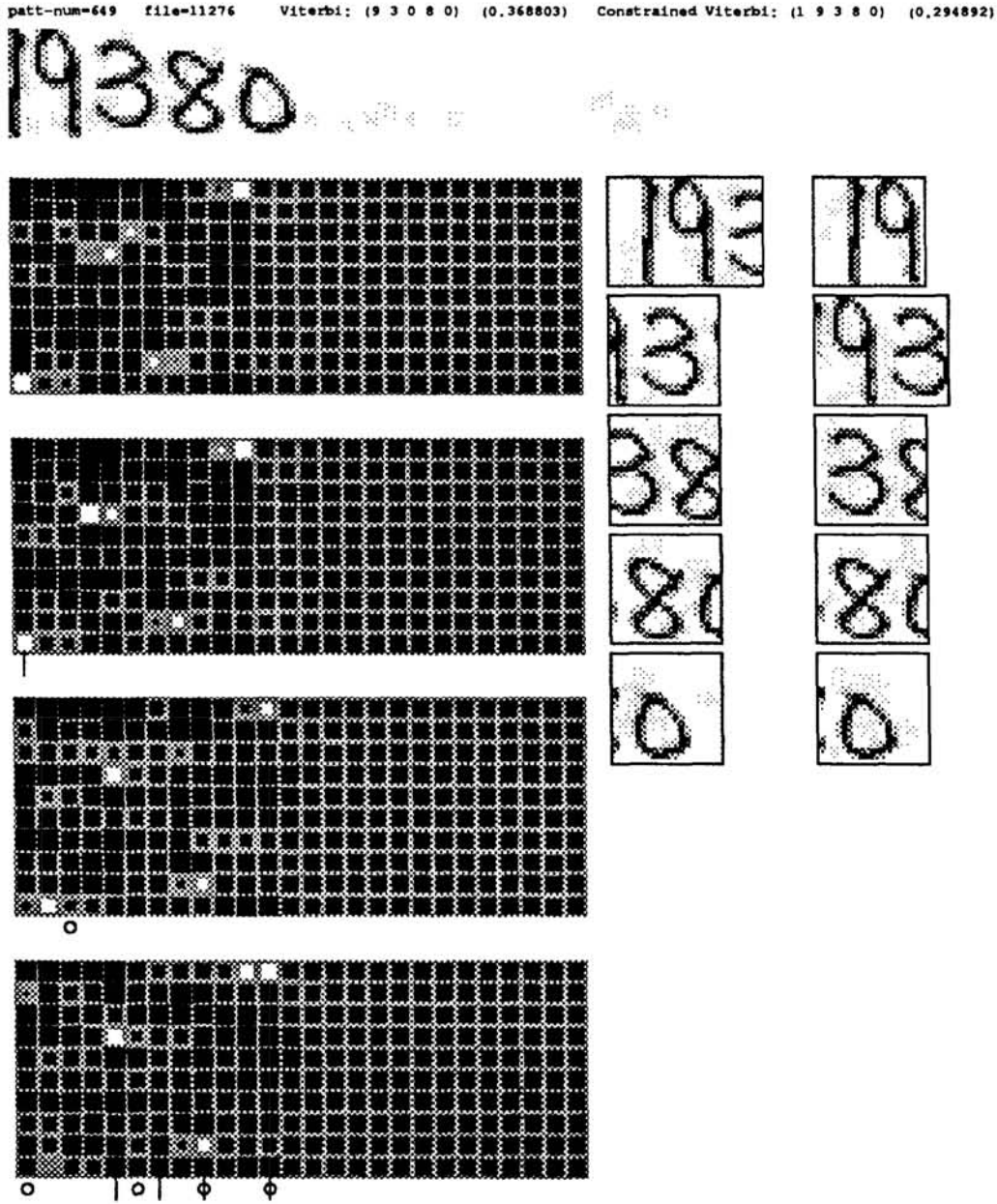

Figure 4: Training using the SDNN/Viterbi. The output vectors chosen by the Viterbi algorithm are marked by a thin line beneath them. The corresponding input regions are shown in the left column. The output vectors chosen by the constrained Viterbi algorithm are marked by small circles and their corresponding input regions are shown to the right. Given the ground truth the system can learn to center on the correct digit.

through all other units except those that are "similar" to ones in the correct set. ("similar" is defined by corresponding to a close center of frame in the input and responding with the correct class).

As one adds more units that have a non zero gradient, each training iteration is more similar to batch-training and is more prone to oscillations. In this case more Newton sweeps are required.

## 5     Results

The current raw recognition rates for the whole 5-digit string are 70% correct from the training set and 66% correct from the test set. Additional interesting statistics are the distribution of the number of correct digits across the whole ZIP code and the recognition rates for each digit's position within the ZIP code. These are presented in the tables shown below.

Table 1: Top: Distribution of test images according to the number of correct single digit classifications out of 5. Bottom: Rates of single digit classification according to position. Digits on the edges are classified more easily since one edge is predetermined.

| Number of digits correct | Percent of cases |
|---|---|
| 5 | 66.3 |
| 4 | 19.7 |
| 3 | 7.2 |
| 2 | 4.7 |
| 1 | 1.4 |
| 0 | 0.7 |

| Digit position | Percent correct |
|---|---|
| 1st | 92 |
| 2nd | 87 |
| 3rd | 87 |
| 4th | 86 |
| 5th | 90 |

## 6     Conclusions and Future Work

The SDNN combined with the Viterbi algorithm learns to recognize strings of handwritten digits by "centering" on the individual digits in the string. This is similar in concept to other work in speech (Haffner, Franzini and Waibel, 1991) but differs from (Keeler, Rumelhart and Leow, 1991), where no alignment procedure is used.

The current recognition rates are still lower than our best system that uses pixel projection information to guide a recognition based segmenter. The SDNN is much faster and lends itself to parallel hardware. Possible improvements to the architecture may be:

- Modified constraints on the segmentation rules of the feature layer.
- Applying the Viterbi algorithm in the vertical direction as well might overcome problems due to height variance.
- It might be too hard to segment using local information only ; one might try using global information, such as pixel projection or recognizing doublets or triplets.

Though there is still considerable work to be done in order to reach state-of-the-art recognition levels, we believe that this type of approach is the correct direction for future image processing applications. Applying recognition based segmentation at the line, word and character level on high feature maps is necessary in order to achieve fast processing while exploring a large set of possible interpretations.

## Acknowledgements

Support of this work by the Technology Resource Department of the U.S. Postal Service under Task Order 104230-90-C-2456 is gratefully acknowledged.

## Footnotes

*Author's current address: Department of Computer Science, Stanford University, Stanford, CA 94305.

# References

Bridle, J. S. (1989). Probabilistic Interpretation of Feedforward Classification Network Outputs with Relationships to Statistical Pattern Recognition. In Fogelman-Soulie, F. and Hérault, J., editors, *Neuro-computing: algorithms, architectures and applications*. Springer-Verlag.

Franzini, M., Lee, K. F., and Waibel, A. (1990). Connectionist Viterbi Training: A New Hybrid Method For Continuous Speech Recognition. In *Proceedings ICASSP 90*, pages 425–428. IEEE.

Haffner, P., Franzini, M., and Waibel, A. (1991). Integrating Time Alignment and Neural Networks for High Performance Continuous Speech Recognition. In *Proceedings ICASSP 91*. IEEE.

Keeler, J. D., Rumelhart, D. E., and Leow, W. (1991). Integrated Segmentation and Recognition of Handwritten-Printed Numerals. In Lippman, Moody, and Touretzky, editors, *Advances in Neural Information Processing Systems*, volume 3. Morgan Kaufman.

Lang, K. J. and Hinton, G. E. (1988). A Time Delay Neural Network Architecture for Speech Recognition. Technical Report CMU-cs-88-152, Carnegie-Mellon University, Pittsburgh PA.

Le Cun, Y., Matan, O., Boser, B., Denker, J. S., Henderson, D., Howard, R. E., Hubbard, W., Jackel, L. D., and Baird, H. S. (1990). Handwritten Zip Code Recognition with Multilayer Networks. In *Proceedings of the 10th International Conference on Pattern Recognition*. IEEE Computer Society Press.

Matan, O., Bromley, J., Burges, C. J. C., Denker, J. S., Jackel, L. D., Le Cun, Y., Pednault, E. P. D., Satterfield, W. D., Stenard, C. E., and Thompson, T. J. (1991). Reading Handwritten Digits: A ZIP code Recognition System (To appear in COMPUTER).

Rabiner, L. R. (1989). A Tutorial on Hidden Markov Models and Selected Applications in Speech Recognition. *Proceedings of the IEEE*, 77:257–286.